# Classification with Hybrid Generative/Discriminative Models

**Rajat Raina, Yirong Shen, Andrew Y. Ng**
Computer Science Department
Stanford University
Stanford, CA 94305

**Andrew McCallum**
Department of Computer Science
University of Massachusetts
Amherst, MA 01003

## Abstract

Although discriminatively trained classifiers are usually more accurate when labeled training data is abundant, previous work has shown that when training data is limited, generative classifiers can out-perform them. This paper describes a hybrid model in which a high-dimensional subset of the parameters are trained to maximize generative likelihood, and another, small, subset of parameters are discriminatively trained to maximize conditional likelihood. We give a sample complexity bound showing that in order to fit the discriminative parameters well, the number of training examples required depends only on the logarithm of the number of feature occurrences and feature set size. Experimental results show that hybrid models can provide lower test error and can produce better accuracy/coverage curves than either their purely generative or purely discriminative counterparts. We also discuss several advantages of hybrid models, and advocate further work in this area.

## 1 Introduction

Generative classifiers learn a model of the joint probability, $p(x, y)$, of the inputs $x$ and the label $y$, and make their predictions by using Bayes rule to calculate $p(y|x)$, and then picking the most likely label $y$. In contrast, discriminative classifiers model the posterior $p(y|x)$ directly. It has often been argued that for many application domains, discriminative classifiers often achieve higher test set accuracy than generative classifiers (e.g., [6, 4, 14]). Nonetheless, generative classifiers also have several advantages, among them straightforward EM methods for handling missing data, and often better performance when training set sizes are small. Specifically, it has been shown that a simple generative classifier (naive Bayes) outperforms its conditionally-trained, discriminative counterpart (logistic regression) when the amount of available labeled training data is small [11].

In an effort to obtain the best of both worlds, this paper explores a class of hybrid models for supervised learning that are partly generative and partly discriminative. In these models, a large subset of the parameters are trained to maximize the generative, joint probability of the inputs and outputs of the supervised learning task; another, much smaller, subset of the parameters are discriminatively trained to maximize the conditional probability of the outputs given the inputs.

Motivated by an application in text classification as well as a desire to begin by exploring a simple, pure form of hybrid classification, we describe and give results with a "generative-discriminative" pair [11] formed by naive Bayes and logistic regression, and a hybrid al-

gorithm based on both. We also give two natural by-products of the hybrid model: First, a scheme for allowing different partitions of the variables to contribute more or less strongly to the classification decision—for an email classification example, modeling the text in the subject line and message body separately, with learned weights for the relative contributions. Second, a method for improving accuracy/coverage curves of models that make incorrect independence assumptions, such as naive Bayes.

We also prove a sample complexity result showing that the number of training examples needed to fit the discriminative parameters depends only on the logarithm of the vocabulary size and document length. In experimental results, we show that the hybrid model achieves significantly more accurate classification than either its purely generative or purely discriminative counterparts. We also demonstrate that the hybrid model produces class posterior probabilities that better reflect empirical error rates, and as a result produces improved accuracy/coverage curves.

## 2   The Model

We begin by briefly reviewing the multinomial naive Bayes classifier applied to text categorization [10], and then describe our hybrid model and its relation to logistic regression.

Let $\mathcal{Y} = \{0, 1\}$ be the set of possible labels for a document classification task, and let $\mathcal{W} = \{w_1, w_2, \ldots, w_{|\mathcal{W}|}\}$ be a dictionary of words. A document of $N$ words is represented by a vector $X = (X_1, X_2, \ldots, X_N)$ of length $N$. The $i$th word in the document is $X_i \in \mathcal{W}$. Note that $N$ can vary for different documents. The multinomial naive Bayes model assumes that the label $Y$ is chosen from some prior distribution $P(Y = \cdot)$, the length $N$ is drawn from some distribution $P(N = \cdot)$ independently of the label, and each word $X_i$ is drawn independently from some distribution $P(W = \cdot|Y)$ over the dictionary. Thus, we have:[1]

$$P(X = x, Y = y) = P(Y = y)P(N = n) \prod_{i=1}^{n} P(W = x_i|Y = y). \qquad (1)$$

Since the length $n$ of the document does not depend on the label and therefore does not play a significant role, we leave it out of our subsequent derivations.

The parameters in the naive Bayes model are $\hat{P}(Y)$ and $\hat{P}(W|Y)$ (our estimates of $P(Y)$ and $P(W|Y)$). They are set to maximize the joint (penalized) log-likelihood of the $x$ and $y$ pairs in a labeled training set, $M = \{(x^{(i)}, y^{(i)})\}_{i=1}^{m}$. Let $n^{(i)}$ be the length of document $x^{(i)}$. Specifically, for any $k \in \{0, 1\}$, we have:

$$\hat{P}(Y = k) \quad = \quad \frac{1}{m} \sum_{i=1}^{m} 1\{y^{(i)} = k\} \qquad (2)$$

$$\hat{P}(W = w_l|Y = k) \quad = \quad \frac{\sum_{i=1}^{m} \sum_{j=1}^{n^{(i)}} 1\{x_j^{(i)} = w_l, \, y^{(i)} = k\} + 1}{\sum_{i=1}^{m} n^{(i)} 1\{y^{(i)} = k\} + |\mathcal{W}|}, \qquad (3)$$

where $1\{\cdot\}$ is the indicator function ($1\{\text{True}\} = 1$, $1\{\text{False}\} = 0$), and we have applied Laplace (add-one) smoothing in obtaining the estimates of the word probabilities. Using Bayes rule, we obtain the estimated class posterior probabilities for a new document $x$ as:

$$\hat{P}(Y = 1|X = x) = \frac{\hat{P}(X=x|Y=1)\hat{P}(Y=1)}{\sum_{y \in \mathcal{Y}} \hat{P}(X=x|Y=y)\hat{P}(Y=y)}$$

where

$$\hat{P}(X = x|Y = y) = \prod_{i=1}^{n} \hat{P}(W = x_i|Y = y). \qquad (4)$$

The predicted class for the new document is then simply $\arg\max_{y \in \mathcal{Y}} \hat{P}(Y = y|X = x)$.

In many text classification applications, the documents involved consist of several disjoint regions that may have different dependencies with the document label. For example, a USENET news posting includes both a subject region and a message body region.[2] Because

of the strong assumptions used by naive Bayes, it treats the words in the different regions of a document in exactly the same way, ignoring the fact that perhaps words in a particular region (such as words in the subject) might be more "important." Further, it also tends to allow the words in the longer region to dominate. (Explained below.)

In the sequel, we assume that every input document $X$ can be naturally divided into $R$ regions $X^1, X^2, \ldots, X^R$. Note that $R$ can be one. The regions are of variable lengths $N_1, N_2, \ldots, N_R$. For the sake of conciseness and clarity, in the following discussion we will focus on the case of $R = 2$ regions, the generalization offering no difficulties. Thus, the document probability in Equation (4) is now replaced with:

$$\hat{P}(X = x|Y = y) = \hat{P}(X^1 = x^1|Y = y)\hat{P}(X^2 = x^2|Y = y) \tag{5}$$

$$= \prod_{i=1}^{n_1} \hat{P}(W = x_i^1|Y = y) \prod_{i=1}^{n_2} \hat{P}(W = x_i^2|Y = y) \tag{6}$$

Here, $x_i^j$ denotes the $i$th word in the $j$th region. Naive Bayes will predict $y = 1$ if:

$$\sum_{i=1}^{n_1} \log \hat{P}(W = x_i^1|Y = 1) + \sum_{i=1}^{n_2} \log \hat{P}(W = x_i^2|Y = 1) + \log \hat{P}(Y = 1) \geq$$

$$\sum_{i=1}^{n_1} \log \hat{P}(W = x_i^1|Y = 0) + \sum_{i=1}^{n_2} \log \hat{P}(W = x_i^2|Y = 0) + \log \hat{P}(Y = 0)$$

and predict $y = 0$ otherwise. In an email or USENET news classification problem, if the first region is the subject, and the second region is the message body, then $n_2 \gg n_1$, since message bodies are usually much longer than subjects. Thus, in the equation above, the message body contributes to many more terms in both the left and right sides of the summation, and the result of the "$\geq$" test will be largely determined by the message body (with the message subject essentially ignored or otherwise having very little effect).

Given the importance and informativeness of message subjects, this suggests that we might obtain better performance than the basic naive Bayes classifier by considering a modified algorithm that assigns different "weights" to different regions, and normalizes for region lengths. Specifically, consider making a prediction using the modified inequality test:

$$\frac{\theta_1}{n_1} \sum_{i=1}^{n_1} \log \hat{P}(W = x_i^1|Y = 1) + \frac{\theta_2}{n_2} \sum_{i=1}^{n_2} \log \hat{P}(W = x_i^2|Y = 1) + \log \hat{P}(Y = 1) \geq$$

$$\frac{\theta_1}{n_1} \sum_{i=1}^{n_1} \log \hat{P}(W = x_i^1|Y = 0) + \frac{\theta_2}{n_2} \sum_{i=1}^{n_2} \log \hat{P}(W = x_i^2|Y = 0) + \log \hat{P}(Y = 0)$$

Here, the vector of parameters $\theta = (\theta_1, \theta_2)$ controls the relative "weighting" between the message subjects and bodies, and will be fit discriminatively. Specifically, we will model the class posteriors, which we denote by $\hat{P}_\theta$ to make explicit the dependence on $\theta$, as:[3]

$$\hat{P}_\theta(y|x) = \frac{\hat{P}(y)\hat{P}(x^1|y)^{\frac{\theta_1}{n_1}}\hat{P}(x^2|y)^{\frac{\theta_2}{n_2}}}{\hat{P}(Y=0)\hat{P}(x^1|Y=0)^{\frac{\theta_1}{n_1}}\hat{P}(x^2|Y=0)^{\frac{\theta_2}{n_2}} + \hat{P}(Y=1)\hat{P}(x^1|Y=1)^{\frac{\theta_1}{n_1}}\hat{P}(x^2|Y=1)^{\frac{\theta_2}{n_2}}} \tag{7}$$

We had previously motivated our model as assigning different weights to different parts of the document. A second reason for using this model is that the independence assumptions of naive Bayes are too strong. Specifically, with a document of length $n$, the classifier "assumes" that it has $n$ completely independent pieces of evidence supporting its conclusion about the document's label. Putting $n_r$ in the denominator of the exponent as a normalization factor can be viewed as a way of counteracting the overly strong independence assumptions.[4]

After some simple manipulations, we obtain the following expression for $\hat{P}_\theta(Y = 1|x)$:

$$\hat{P}_\theta(Y = 1|x) = \frac{1}{1 + \exp(-a - \theta_1 b_1 - \ldots - \theta_R b_R)} \tag{8}$$

where $a = \log \frac{\hat{P}(Y=1)}{\hat{P}(Y=0)}$ and $b_r = \frac{1}{n_r}(\log \frac{\hat{P}(x^r|Y=1)}{\hat{P}(x^r|Y=0)})$. With this expression for $\hat{P}_\theta(y|x)$, we see that it is very similar to the form of the class posteriors used by logistic regression, the

only difference being that in this case $a$ is a constant calculated from the estimated class priors. To make the parallel to logistic regression complete, we define $b_0 = 1$, redefine $\theta$ as $\theta = (\theta_0, \theta_1, \theta_2)$, and define a new class posterior

$$\hat{P}_\theta(Y = 1|x) = \frac{1}{1+\exp(-\theta^T b)} \tag{9}$$

Throughout the derivation, we had assumed that the parameters $\hat{P}(x|y)$ were fit generatively as in Equation (3) (and $b$ is in turn derived from these parameters as described above). It therefore remains only to specify how $\theta$ is chosen. One method would be to pick $\theta$ by maximizing the conditional log-likelihood of the training set $M = \{x^{(i)}, y^{(i)}\}_{i=1}^m$:

$$\theta = \arg\max_{\theta'} \sum_{i=1}^m \log \hat{P}_{\theta'}(y^{(i)}|x^{(i)}) \tag{10}$$

However, the word generation probabilities that were used to calculate $b$ were also trained from the training set $M$. This procedure therefore fits the parameters $\theta$ to the training data, using "features" $b$ that were also fit to the data. This leads to a biased estimator. Specifically, since what we care about is the generalization performance of the algorithm, a better method is to pick $\theta$ to maximize the log-likelihood of data that wasn't used to calculate the "features" $b$, because when we see a test example, we will not have had the luxury of incorporating information from the test example into the $b$'s (cf. [15, 12]). This leads to the following "leave-one-out" strategy of picking $\theta$:

$$\theta = \arg\max_{\theta'} \sum_{i=1}^m \log \hat{P}_{\theta', -i}(y^{(i)}|x^{(i)}), \tag{11}$$

where $\hat{P}_{\hat{\theta}, -i}(y^{(i)}|x^{(i)})$ is as given in Equation (9), except that each $b_r$ is computed from word generation probabilities that were estimated with the $i$th example of the training set held out. We note that optimizing this objective to find $\theta$ is still the same optimization problem as in logistic regression, and hence is convex and can be solved efficiently. Further, the word generation probabilities with the $i$th example left out can also be computed efficiently.[5]

The predicted label for a new document under this method is $\arg\max_{y \in \mathcal{Y}} \hat{P}_\theta(y|x)$. We call this method the *normalized hybrid* algorithm. For the sake of comparison, we will also consider an algorithm in which the exponents in Equation (7) are not normalized by $n_r$. In other words, we replace $\theta_r/n_r$ there by just $\theta_r$. We refer to this latter method as the *unnormalized hybrid* algorithm.

## 3 Experimental Results

We now describe the results of experiments testing the effectiveness of our methods. All experiments were run using pairs of newsgroups from the 20newsgroups dataset [8] of USENET news postings. When parsing this data, we skipped everything in the USENET headers except the subject line; numbers and email addresses were replaced by special tokens NUMBER and EMAILADDR; and tokens were formed after stemming.

In each experiment, we compare the performance of the basic naive Bayes algorithm with that of the normalized hybrid algorithm and logistic regression with Gaussian priors on the parameters. We used logistic regression with word-counts in the feature vectors (as in [6]), which forms a discriminative-generative pair with multinomial naive Bayes. All results reported in this section are averages over 10 random train-test splits.

Figure 1 plots learning curves for the algorithms, when used to classify between various pairs of newsgroups. We find that in every experiment, for the training set sizes considered, the normalized hybrid algorithm with $R = 2$ has test error that is either the lowest or very near the lowest among all the algorithms. In particular, it almost always outperforms the

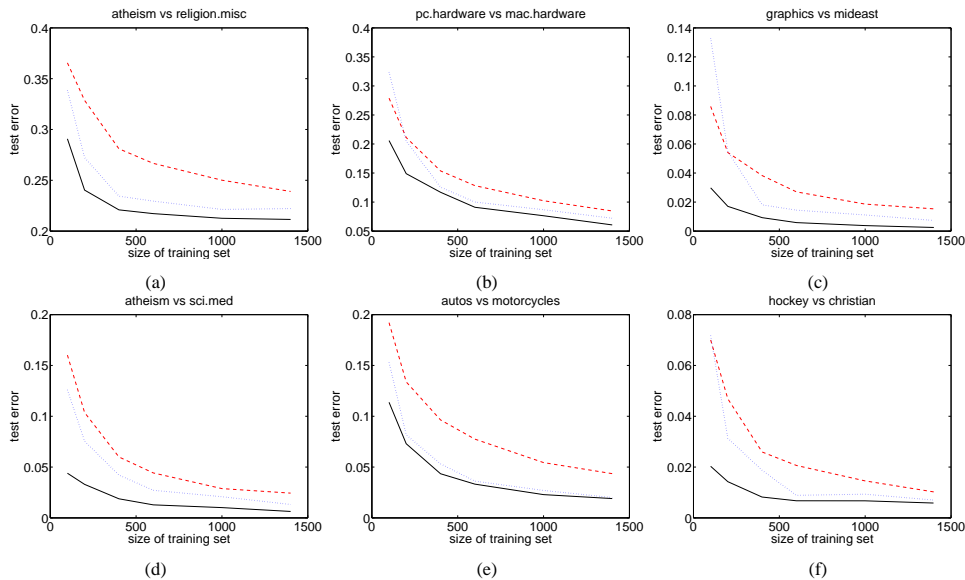

Figure 1: Plots of test error vs training size for several different newsgroup pairs. Red dashed line is logistic regression; blue dotted line is standard naive Bayes; black solid line is the hybrid algorithm. (Colors where available.) (If more training data were available, logistic regression would presumably out-perform naive Bayes; cf. [6, 11].)

basic naive Bayes algorithm. The difference in performance is especially dramatic for small training sets.

Although these results are not shown here, the hybrid algorithm with $R = 2$ (breaking the document into two regions) outperforms $R = 1$. Further, the normalized version of the hybrid algorithm generally outperforms the unnormalized version.

## 4 Theoretical Results

In this section, we give a distribution free uniform convergence bound for our algorithm. Classical learning and VC theory indicates that, given a discriminative model with a small number of parameters, typically only a small amount of training data should be required to fit the parameters "well" [14]. In our model, a large number of parameters $\hat{P}$ are fit generatively, but only a small number (the $\theta$'s) are fit discriminatively. We would like to show that only a small training set is required to fit the discriminative parameters $\theta$.[6] However, standard uniform convergence results do not apply to our problem, because the "features" $b_i$ given to the discriminative logistic regression component also depend on the training set. Further, the $\theta_i$'s are fit using the leave-one-out training procedure, so that every pair of training examples is actually dependent.

For our analysis, we assume the training set of size $m$ is drawn *i.i.d.* from some distribution $\mathcal{D}$ over $\mathcal{X} \times \mathcal{Y}$. Although not necessary, for simplicity we assume that each document has the same total number of words $n = \sum_{i=1}^{R} n_i$, though the lengths of the individual regions may vary. (It also suffices to have an upper- and a lower-bound on document length.) Finally, we also assume that each word occurs at most $C_{max}$ times in a single document, and that the distribution $\mathcal{D}$ from which training examples are drawn satisfies

$\rho_{min} \leq P(Y = 1) \leq 1 - \rho_{min}$, for some fixed $\rho_{min} > 0$.

Note that we do *not* assume that the "naive Bayes assumption" (that words are conditionally independent given the class label) holds. Specifically, even when the naive Bayes assumption does not hold, the naive Bayes *algorithm* (as well as our hybrid algorithm) can still be applied, and our results apply to this setting.

Given a set $M$ of $m$ training examples, for a particular setting of the parameter $\theta$, the expected log likelihood of a randomly drawn test example is:

$$\varepsilon^M(\theta) \quad = \quad E_{(x,y)\sim\mathcal{D}} \log \hat{P}_\theta(y|x) \tag{12}$$

where $\hat{P}_\theta$ is the probability model trained on $M$ as described in the previous section, using parameters $\hat{P}$ fit to the entire training set. Our algorithm uses a leave-one-out estimate of the true log likelihood; we call this the leave-one-out log likelihood:

$$\hat{\varepsilon}^M_{-1}(\theta) \quad = \quad \frac{1}{m} \sum_{i=1}^m \log \hat{P}_{\theta,-i}(y^{(i)}|x^{(i)}) \tag{13}$$

where $\hat{P}_{\theta,-i}$ represents the probability model trained with the $i$th example left out.

We would like to choose $\theta$ to maximize $\varepsilon^M$, but we do not know $\varepsilon^M$. Now, it is well-known that if we have some estimate $\hat{\varepsilon}$ of a generalization error measure $\varepsilon$, and if $|\hat{\varepsilon}(\theta) - \varepsilon(\theta)| \leq \epsilon$ for all $\theta$, then optimizing $\hat{\varepsilon}$ will result in a value for $\theta$ that comes within $2\epsilon$ of the best possible value for $\varepsilon$ [14]. Thus, in order to show that optimizing $\hat{\varepsilon}^M_{-1}$ is a good "proxy" for optimizing $\varepsilon^M$, we only need to show that $\hat{\varepsilon}^M_{-1}(\theta)$ is uniformly close to $\varepsilon^M(\theta)$. We have:

**Theorem 1** *Under the previous set of assumptions, in order to ensure that with probability at least $1 - \delta$, we have $|\varepsilon^M(\theta) - \hat{\varepsilon}^M_{-1}(\theta)| < \epsilon$ for all parameters $\theta$ such that $||\theta||_\infty \leq \eta$, it suffices that $m = O(\text{poly}(1/\delta, 1/\epsilon, \log n, \log |\mathcal{W}|, R, \eta)^R)$.*

The full proof of this result is fairly lengthy, and is deferred to the full version of this paper [13]. From the theorem, the number of training examples $m$ required to fit the $\theta$ parameters (under the fairly standard regularity condition that $\theta$ be bounded) depends only on the logarithms of the document length $n$ and the vocabulary size $|\mathcal{W}|$. In our bound, there is an exponential dependence on $R$; however, from our experience, $R$ does not need to be too large for significantly improved performance. In fact, our experimental results demonstrate good performance for $R = 2$.

## 5 Calibration Curves

We now consider a second application of these ideas, to a text classification setting where the data is not naturally split into different regions (equivalently, where $R = 1$). In this setting we cannot use the "reweighting" power of the hybrid algorithm to reduce classification error. But, we will see that, by giving better class posteriors, our method still gives improved performance as measured on accuracy/coverage curves.

An accuracy/coverage curve shows the accuracy (fraction correct) of a classifier if it is asked only to provide $x\%$ coverage—that is, if it is asked only to label the $x\%$ of the test data on which it is most confident. Accuracy/coverage curves towards the upper-right of the graph mean high accuracy even when the coverage is high, and therefore good performance. Accuracy value at coverage 100% is just the normal classification error. In settings where both human and computer label documents, accuracy/coverage curves play a central role in determining how much data has to be labeled by humans. They are also indicative of the quality of a classifier's class posteriors, because a classifier with better class posteriors would be able to better judge which $x\%$ of the test data it should be most confident on, and achieve higher accuracy when it chooses to label that $x\%$ of the data.

Figure 2 shows accuracy/coverage curves for classifying several pairs of newsgroups from the 20newsgroups dataset. Each plot is obtained by averaging the results of ten 50%/50% random train/test splits. The normalized hybrid algorithm ($R = 1$) does significantly better than naive Bayes, and has accuracy/coverage curves that are higher almost everywhere.

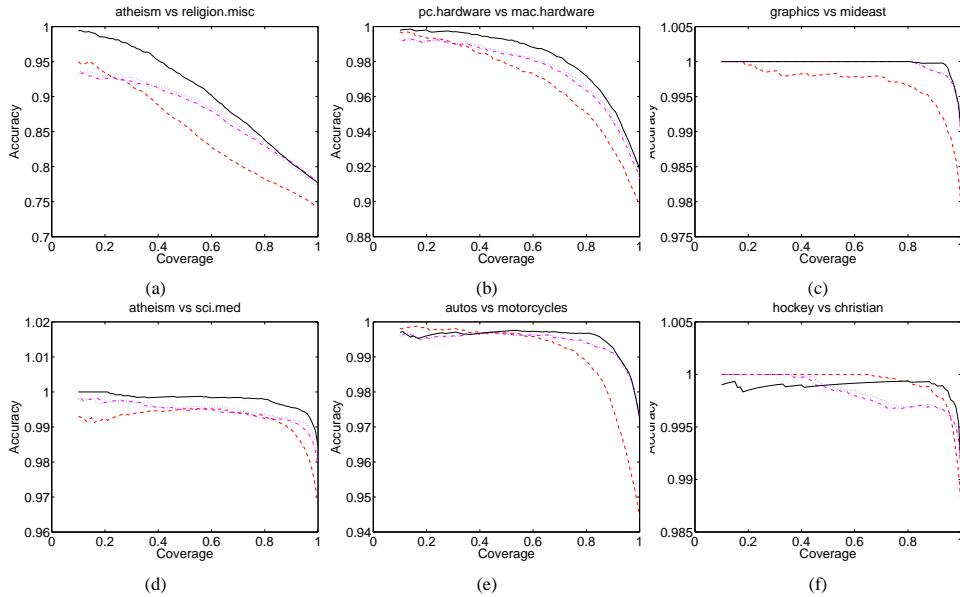

Figure 2: Accuracy/Coverage curves for different newsgroups pairs. Black solid line is our normalized hybrid algorithm with $R = 1$; magenta dash-dot line is naive Bayes; blue dotted line is unnormalized hybrid, and red dashed line is logistic regression. (Colors where available.)

For example, in Figure 2a, the normalized hybrid algorithm with $R = 1$ has a coverage of over 40% at 95% accuracy, while naive Bayes' coverage is 0 for the same accuracy. Also, the unnormalized algorithm has performance about the same as naive Bayes. Even in examples where the various algorithms have comparable overall test error, the normalized hybrid algorithm has significantly better accuracy/coverage.

## 6    Discussion and Related Work

This paper has described a hybrid generative/discriminative model, and presented experimental results showing that a simple hybrid model can perform better than either its purely generative or discriminative counterpart. Furthermore, we showed that in order to fit the parameters $\theta$ of the model, only a small number of training examples is required.

There have been a number of previous efforts to modify naive Bayes to obtain more empirically accurate posterior probabilities. Lewis and Gale [9] use logistic regression to re-calibrate naive Bayes posteriors in an active learning task. Their approach is similar to the lower-performing *unnormalized* version of our algorithm, with only one region. Bennett [1] studies the problem of using asymmetric parametric models to obtain high quality probability estimates from the scores outputted by text classifiers such as naive Bayes. Zadrozny and Elkan [16] describe a simple non-parametric method for calibrating naive Bayes probability estimates. While these methods can obtain good class posteriors, we note that in order to obtain better accuracy/coverage, it is not sufficient to take naive Bayes' output $p(y|x)$ and find a monotone mapping from that to a set of hopefully better class posteriors (e.g., [16]). Specifically, in order to obtain better accuracy/coverage, it is also important to *rearrange* the confidence orderings that naive Bayes gives to documents (which our method does because of the normalization).

Jaakkola and Haussler [3] describe a scheme in which the kernel for a discriminative classifier is extracted from a generative model. Perhaps the closest to our work, however, is

the commonly-used, simple "reweighting" of the language model and acoustic model in speech recognition systems (e.g., [5]). Each of the two models is trained generatively; then a single weight parameter is set using hold-out cross-validation.

In related work, there are also a number of theoretical results on the quality of leave-one-out estimates of generalization error. Some examples include [7, 2]. (See [7] for a brief survey.) Those results tend to be for specialized models or have strong assumptions on the model, and to our knowledge do not apply to our setting, in which we are also trying to fit the parameters $\theta$.

In closing, we have presented one hybrid generative/discriminative algorithm that appears to do well on a number of problems. We suggest that future research in this area is poised to bear much fruit. Some possible future work includes: automatically determining which parameters to train generatively and which discriminatively; training methods for more complex models with latent variables, that require EM to estimate both sets of parameters; methods for taking advantage of the hybrid nature of these models to better incorporate domain knowledge; handling missing data; and support for semi-supervised learning.

**Acknowledgments.** We thank Dan Klein, David Mulford and Ben Taskar for helpful conversations. Y. Shen is supported by an NSF graduate fellowship. This work was also supported by the Department of the Interior/DARPA under contract number NBCHD030010, and NSF grant #IIS-0326249.

## Footnotes

[1]We adopt the notational convention that upper-case is used to denote random variables, and lower-case is used to denote particular values taken by the random variables.

[2]Other possible text classification examples include: Emails consisting of subject and body; technical papers consisting of title, abstract, and body; web pages consisting of title, headings, and body.

[3]When there is no risk of ambiguity, we will sometimes replace $P(X = x|Y = y)$, $P(Y = y|X = x)$, $P(W = x_i|Y = y)$, etc. with $P(x|y)$, $P(y|x)$, $P(x_i|y)$.

[4]$\theta_r$ can also be viewed as an "effective region length" parameter, where we assume that region $r$ of the document can be treated as only $\theta_r$ independent pieces of observation. For example, note that if each region $r$ of the document has $\theta_r$ words exactly, then this model reduces to naive Bayes.

[5]Specifically, by precomputing the numerator and denominator of Equation (3), we can later remove any example by subtracting out the terms in the numerator and denominator corresponding to that example.

[6]For a result showing that naive Bayes' generatively fit parameters (albeit one using a different event model) converge to their population (asymptotic) values after a number of training examples that depends logarithmically on the size of the number of features, also see [11].

# References

[1] Paul N. Bennett. Using asymmetric distributions to improve text classifier probability estimates. In *Proceedings of SIGIR-03, 26th ACM International Conference on Research and Development in Information Retrieval*, 2003.

[2] Luc P. Devroye and T. J. Wagner. Distribution-free performance bounds for potential function rules. *IEEE Transactions on Information Theory*, 5, September 1979.

[3] T. Jaakkola and D. Haussler. Exploiting generative models in discriminative classifiers. In *Advances in Neural Information Processing Systems 11*, 1998.

[4] T. Jebara and A. Pentland. Maximum conditional likelihood via bound maximization and the cem algorithm. In *Advances in Neural Information Processing Systems 11*, 1998.

[5] D. Jurafsky and J. Martin. *Speech and language processing*. Prentice Hall, 2000.

[6] John Lafferty Kamal Nigam and Andrew McCallum. Using maximum entropy for text classification. In *IJCAI-99 Workshop on Machine Learning for Information Filtering*, 1999.

[7] Michael Kearns and Dana Ron. Algorithmic stability and sanity-check bounds for leave-one-out cross-validation. *Computational Learning Theory*, 1997.

[8] Ken Lang. Newsweeder: learning to filter netnews. In *Proceedings of the Ninth European Conference on Machine Learning*, 1997.

[9] David D. Lewis and William A. Gale. A sequential algorithm for training text classifiers. In *Proceedings of SIGIR-94, 17th ACM International Conference on Research and Development in Information Retrieval*, 1994.

[10] Andrew McCallum and Kamal Nigam. A comparison of event models for naive bayes text classification. In *AAAI-98 Workshop on Learning for Text Categorization*, 1998.

[11] Andrew Y. Ng and Michael I. Jordan. On discriminative vs. generative classifiers: a comparison of logistic regression and naive bayes. In *NIPS 14*, 2001.

[12] John C. Platt. Probabilistic outputs for support vector machines and comparisons to regularized likelihood methods. In A. Smola, P. Bartlett, B. Scholkopf, and D. Schuurmans, editors, *Advances in Large Margin Classifiers*. MIT Press, 1999.

[13] R. Raina, Y. Shen, A. Y. Ng, and A. McCallum. Classification with hybrid generative/discriminative models. http://www.cs.stanford.edu/~rajatr/nips03.ps, 2003.

[14] V. N. Vapnik. *Statistical Learning Theory*. John Wiley & Sons, 1998.

[15] David H. Wolpert. Stacked generalization. *Neural Networks*, 5(2):241–260, 1992.

[16] Bianca Zadrozny and Charles Elkan. Obtaining calibrated probability estimates from decision trees and naive bayesian classifiers. In *ICML '01*, 2001.
